# Infinite Relational Modeling of Functional Connectivity in Resting State fMRI

**Morten Mørup**
Section for Cognitive Systems
DTU Informatics
Technical University of Denmark
mm@imm.dtu.dk

**Kristoffer Hougaard Madsen**
Danish Research Centre for Magnetic Resonance
Copenhagen University Hospital Hvidovre
khm@drcmr.dk

**Anne Marie Dogonowski**
Danish Research Centre for Magnetic Resonance
Copenhagen University Hospital Hvidovre
annemd@drcmr.dk

**Hartwig Siebner**
Danish Research Centre for Magnetic Resonance
Copenhagen University Hospital Hvidovre
hartwig.siebner@drcmr.dk

**Lars Kai Hansen**
Section for Cognitive Systems
DTU Informatics
Technical University of Denmark
lkh@imm.dtu.dk

## Abstract

Functional magnetic resonance imaging (fMRI) can be applied to study the functional connectivity of the neural elements which form complex network at a whole brain level. Most analyses of functional resting state networks (RSN) have been based on the analysis of correlation between the temporal dynamics of various regions of the brain. While these models can identify coherently behaving groups in terms of correlation they give little insight into how these groups interact. In this paper we take a different view on the analysis of functional resting state networks. Starting from the definition of resting state as functional coherent groups we search for functional units of the brain that communicate with other parts of the brain in a coherent manner as measured by mutual information. We use the infinite relational model (IRM) to quantify functional coherent groups of resting state networks and demonstrate how the extracted component interactions can be used to discriminate between functional resting state activity in multiple sclerosis and normal subjects.

## 1 Introduction

Neuronal elements of the brain constitute an intriguing complex network [4]. Functional magnetic resonance imaging (fMRI) can be applied to study the functional connectivity of the neural elements which form this complex network at a whole brain level. It has been suggested that fluctuations in the blood oxygenation level-dependent (BOLD) signal during rest reflecting the neuronal baseline activity of the brain correspond to functionally relevant networks [9, 3, 19].

Most analysis of functional resting state networks (RSN) have been based on the analysis of correlation between the temporal dynamics of various regions of the brain either assessed by how well voxels correlate with the signal from predefined regions (so-called) seeds [3, 24] or through unsupervised multivariate approaches such as independent component analysis (ICA) [10, 9]. While

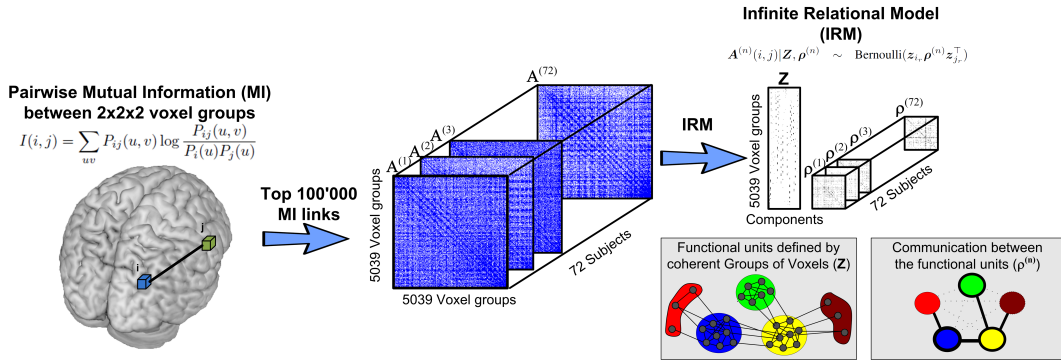

Figure 1: The proposed framework. All pairwise mutual information (MI) are calculated between the 2x2x2 group of voxels for each subjects resting state fMRI activity. The graph of pairwise mutual information is thresholded such that the top 100,000 un-directed links are kept. The graphs are analyzed by the infinite relational model (IRM) assuming the functional units $\boldsymbol{Z}$ are the same for all subjects but their interactions $\boldsymbol{\rho}^{(n)}$ are individual. We will use these extracted interactions to characterize the individuals.

these models identify coherently behaving groups in terms of correlation they give limited insight into how these groups interact. Furthermore, while correlation is optimal for extracting second order statistics it easily fails in establishing higher order interactions between regions of the brain [22, 7].

In this paper we take a different view on the analysis of functional resting state networks. Starting from the definition of resting state as functional coherent groups we search for functional units of the brain that communicate with other parts of the brain in a coherent manner. Consequently, what define functional units are the way in which they interact with the remaining parts of the network. We will consider functional connectivity between regions as measured by mutual information. Mutual information (MI) is well rooted in information theory and given enough data MI can detect functional relations between regions regardless of the order of the interaction [22, 7]. Thereby, resting state fMRI can be represented as a mutual information graph of pairwise relations between voxels constituting a complex network. Numerous studies have analyzed these graphs borrowing on ideas from the study of complex networks [4]. Here common procedures have been to extract various summary statistics of the networks and compare them to those of random networks and these analyses have demonstrated that fMRI derived graphs behave far from random [11, 1, 4]. In this paper we propose to use relational modeling [17, 16, 27] in order to quantify functional coherent groups of resting state networks. In particular, we investigate how this line of modeling can be used to discriminate patients with multiple sclerosis from healthy individuals.

Multiple Sclerosis (MS) is an inflammatory disease resulting in widespread demyelinization of the subcortical and spinal white matter. Focal axonal demyelinization and secondary axonal degeneration results in variable delays or even in disruption of signal transmission along cortico-cortical and cortico-subcortical connections [21, 26]. In addition to the characteristic macroscopic white-matter lesions seen on structural magnetic resonance imaging (MRI), pathology- and advanced MRI-studies have shown demyelinated lesions in cortical gray-matter as well as in white-matter that appear normal on structural MRI [18, 12]. These findings show that demyelination is disseminated throughout the brain affecting brain functional connectivity. Structural MRI gives information about the extent of white-matter lesions, but provides no information on the impact on functional brain connectivity. Given the widespread demyelinization in the brain (i.e., affecting the brain's anatomical and functional 'wiring') MS represents a disease state which is particular suited for relational modeling. Here, relational modeling is able to provide a global view of the communication in the functional network between the extracted functional units. Furthermore, the method facilitates the examination of all brain networks simultaneously in a completely data driven manner. An illustration of the proposed analysis is given in figure 1.

## 2 Methods

**Data:** 42 clinically stable patients with relapsing-remitting (RR) and secondary progressive multiple sclerosis (27 RR; 22 females; mean age: 43.5 years; range 25-64 years) and 30 healthy individuals (15 females; mean age: 42.6 years; range 22-69 years) participated in this cross-sectional study. Patients were neurologically examined and assigned a score according to the EDSS which ranged from 0 to 7 (median EDSS: 4.25; mean disease duration: 14.3 years; range 3-43 years). rs-fMRI was performed with the subjects being at rest and having their eyes closed (3 Tesla Magnetom Trio, Siemens, Erlangen, Germany). We used a gradient echo T2*-weighted echo planar imaging sequence with whole-brain coverage (repetition time: 2490 ms; 3 mm isotropic voxels). The rs-fMRI session lasted 20 min (482 brain volumes). During the scan session the cardiac and respiratory cycles were monitored using a pulse oximeter and a pneumatic belt.

**Preprocessing:** After exclusion of 2 pre-saturation volumes each remaining volume was realigned to the mean volume using a rigid body transformation. The realigned images were then normalized to the MNI template. In order to remove nuisance effects related to residual movement or physiological effects a linear filter comprised of 24 motion related and a total of 60 physiological effects (cardiac, respiratory and respiration volume over time) was constructed [14]. After filtering, the voxel were masked [23] and divided into 5039 voxel groups consisting of $2 \times 2 \times 2$ voxels for the estimation of pairwise MI.

### 2.1 Mutual Information Graphs

The mutual information between voxel groups $i$ and $j$ is given by $I(i,j) = \sum_{uv} P_{ij}(u,v) \log \frac{P_{ij}(u,v)}{P_i(u)P_j(u)}$. Thus, the mutual information hinges on the estimation of the joint density $P_{ij}(u,v)$. Several approaches exists for the estimation of mutual information [25] ranging from parametric to non-parametric methods such as nearest neighbor density estimators [7] and histogram methods. The accuracy of both approaches relies on the number of observations present. We used the histogram approach. We used equiprobable rather than equidistant bins [25] based on 10 percentiles derived from the individual distribution of each voxel group, i.e. $P_i(u) = P_j(v) = \frac{1}{10}$. $P_{ij}(u,v)$ counts the number of co-occurrences of observations from voxels in voxel group $i$ that are at bin $u$ while the corresponding voxels from group $j$ are at bin $v$ at time $t$. As such, we had a total of $8 \cdot 480 = 3840$ samples to populate the 100 bins in the joint histogram. To generate the mutual information graphs for each subject a total of $72 \cdot 5039 \cdot (5039-1)/2 \approx 1$ billion pairwise MI were evaluated. We thresholded each graph keeping the top $100,000$ pairwise MI as links in the graph. As such, each graph had size $5039 \times 5039$ with a total of $200,000$ directed links (i.e. $100,000$ undirected link) which resulted in each graph having link density $\frac{100,000}{5039 \cdot (5039-1)/2} = 0.0079$ while the total number of links was $72 \cdot 100,000 = 7.2$ million links (when counting links only in the one direction).

### 2.2 Infinite Relational Modeling (IRM)

The importance of modeling brain connectivity and interactions is widely recognized in the literature on fMRI [13, 28, 20]. Approaches such as dynamic causal modeling [13], structural equation models [20] and dynamic Bayes nets [28] are normally limited to analysis of a few interactions between known brain regions or predefined regions of interest. The benefits of the current relational modeling approach are that regions are defined in a completely data driven manner while the method establishes interaction at a low computational complexity admitting the analysis of large scale brain networks. Functional connectivity graphs have previously been considered in [6] for the discrimination of schizophrenia. In [24] resting state networks were defined based on normalized graph cuts in order to derive functional units. While normalized cuts are well suited for the separation of voxels into groups of disconnected components the method lacks the ability to consider *coherent interaction* between groups. In [17] the stochastic block model also denoted the relational model (RM) was proposed for the identification of coherent groups of nodes in complex networks. Here, each node $i$ belongs to a class $z_{i_r}$ where $i_r$ denote the $i^{th}$ row of a clustering assignment matrix $\boldsymbol{Z}$, and the probability, $\pi_{ij}$, of a link between node $i$ and $j$ is determined by the class assignments $z_{i_r}$ and $z_{j_r}$ as $\pi_{ij} = \boldsymbol{z}_{i_r} \boldsymbol{\rho} \boldsymbol{z}_{j_r}^{\top}$. Here, $\rho_{k\ell} \in [0,1]$ denotes the probability of generating a link between a node in class $k$ and a node in class $\ell$. Using the Dirichlet process (DP), [16, 27] propose a non-parametric generalization of the model with a potentially infinite number of classes, i.e. the infinite

relational model (IRM). Inference in IRM jointly determines the number of latent classes as well as class assignments and class link probabilities. To our knowledge this is the first attempt to explore the IRM model for fMRI data.

Following [16] we have the following generative model for the infinite relational model

$$
\begin{aligned}
\boldsymbol{Z}|\alpha &\sim \mathrm{DP}(\alpha) \\
\boldsymbol{\rho}^{(n)}(a,b)|\boldsymbol{\beta}_+(a,b), \boldsymbol{\beta}_-(a,b) &\sim \mathrm{Beta}(\boldsymbol{\beta}_+(a,b), \boldsymbol{\beta}_-(a,b)) \\
\boldsymbol{A}^{(n)}(i,j)|\boldsymbol{Z}, \boldsymbol{\rho}^{(n)} &\sim \mathrm{Bernoulli}(\boldsymbol{z}_{i_r}\boldsymbol{\rho}^{(n)}\boldsymbol{z}_{j_r}^\top)
\end{aligned}
$$

As such an entitys tendency to participate in relations is determined solely by its cluster assignment in $\boldsymbol{Z}$. Since the prior on the elements of $\boldsymbol{\rho}$ is conjugate the resulting integral $P(\boldsymbol{A}^{(n)}|\boldsymbol{Z}, \boldsymbol{\beta}_+, \boldsymbol{\beta}_-) = \int P(\boldsymbol{A}^{(n)}|\boldsymbol{\rho}^{(n)}, \boldsymbol{Z})P(\boldsymbol{\rho}^{(n)}|\boldsymbol{\beta}_+, \boldsymbol{\beta}_-)d\boldsymbol{\rho}^{(n)}$ has an analytical solution such that

$$
\begin{aligned}
P(\boldsymbol{A}^{(n)}|\boldsymbol{Z}, \boldsymbol{\beta}_+, \boldsymbol{\beta}_-) &= \prod_{a \geq b} \frac{\mathrm{Beta}(\boldsymbol{M}_+^{(n)}(a,b) + \boldsymbol{\beta}_+(a,b), \boldsymbol{M}_-^{(n)}(a,b) + \boldsymbol{\beta}_-(a,b))}{\mathrm{Beta}(\boldsymbol{\beta}_+(a,b), \boldsymbol{\beta}_-(a,b))}, \\
\boldsymbol{M}_+^{(n)}(a,b) &= (1 - \tfrac{1}{2}\delta_{a,b})\boldsymbol{z}_a^\top(\boldsymbol{A}^{(n)} + \boldsymbol{A}^{(n)^\top})\boldsymbol{z}_b \\
\boldsymbol{M}_-^{(n)}(a,b) &= (1 - \tfrac{1}{2}\delta_{a,b})\boldsymbol{z}_a^\top(\boldsymbol{e}\boldsymbol{e}^\top - \boldsymbol{I})\boldsymbol{z}_b - \boldsymbol{M}_+^{(n)}(a,b)
\end{aligned}
$$

$\boldsymbol{M}_+^{(n)}(a,b)$ is the number of links between functional units $a$ and $b$ whereas $\boldsymbol{M}_-^{(n)}(a,b)$ is the number of non-links between functional unit $a$ and $b$ when disregarding links between a node and itself. $\boldsymbol{e}$ is a vector of length $J$ with ones in all entries where $J$ is the number of voxel groups.

We will assume that the graphs are independent over subjects such that

$$
P(\boldsymbol{A}^{(1)}, \ldots, \boldsymbol{A}^{(N)}|\boldsymbol{Z}, \boldsymbol{\beta}_+, \boldsymbol{\beta}_-) = \prod_n \prod_{a \geq b} \frac{\mathrm{Beta}(\boldsymbol{M}_+^{(n)}(a,b) + \boldsymbol{\beta}_+(a,b), \boldsymbol{M}_-^{(n)}(a,b) + \boldsymbol{\beta}_-(a,b))}{\mathrm{Beta}(\boldsymbol{\beta}_+(a,b), \boldsymbol{\beta}_-(a,b))}.
$$

As a result, the posterior likelihood is given by

$$
P(\boldsymbol{Z}|\boldsymbol{A}^{(1)}, \ldots, \boldsymbol{A}^{(N)}, \boldsymbol{\beta}_+, \boldsymbol{\beta}_-, \alpha) \propto \left(\prod_n P(\boldsymbol{A}^{(n)}|\boldsymbol{Z}, \boldsymbol{\beta}_+, \boldsymbol{\beta}_-)\right)P(\boldsymbol{Z}|\alpha) =
$$

$$
\left(\prod_n \prod_{a \geq b} \frac{\mathrm{Beta}(\boldsymbol{M}_+^{(n)}(a,b) + \boldsymbol{\beta}_+(a,b), \boldsymbol{M}_-^{(n)}(a,b) + \boldsymbol{\beta}_-(a,b))}{\mathrm{Beta}(\boldsymbol{\beta}_+(a,b), \boldsymbol{\beta}_-(a,b))}\right) \cdot \left(\alpha^D \frac{\Gamma(\alpha)}{\Gamma(J+\alpha)} \prod_a \Gamma(n_a)\right).
$$

Where $D$ is the number of expressed functional units and $n_a$ the number of voxel groups assigned to functional unit $a$. The expected value of $\boldsymbol{\rho}^{(n)}$ is given by
$\langle \boldsymbol{\rho}^{(n)}(a,b)\rangle = \frac{\boldsymbol{M}_+^{(n)}(a,b) + \boldsymbol{\beta}_+(a,b)}{\boldsymbol{M}_+^{(n)}(a,b) + \boldsymbol{M}_-^{(n)}(a,b) + \boldsymbol{\beta}_+(a,b) + \boldsymbol{\beta}_-(a,b)}$.

**MCMC Sampling the IRM model:** As proposed in [16] we use a Gibbs sampling scheme in combination with split-merge sampling [15] for the clustering assignment matrix $\boldsymbol{Z}$. We used the split-merge sampling procedure proposed in [15] with three restricted Gibbs sampling sweeps. We initialized the restricted Gibbs sampler by the sequential allocation procedure proposed in [8]. For the MCMC sampling, the posterior likelihood for a node assignment given the assignment of the remaining nodes is needed both for the Gibbs sampler as well as for calculating the split-merge acceptance ratios [15].

$$
P(\boldsymbol{z}_{ia} = 1|\boldsymbol{Z}\backslash\boldsymbol{z}_{i_r}, \boldsymbol{A}^{(1)}, \ldots, \boldsymbol{A}^{(N)}) \propto
\begin{cases}
m_a \prod_n \prod_b \frac{\mathrm{Beta}(\boldsymbol{M}_+^{(n)}(a,b) + \boldsymbol{\beta}_+(a,b), \boldsymbol{M}_-^{(n)}(a,b) + \boldsymbol{\beta}_-(a,b))}{\mathrm{Beta}(\boldsymbol{\beta}_+(a,b), \boldsymbol{\beta}_-(a,b))} & \text{if } m_a > 0 \\
\alpha \prod_n \prod_b \frac{\mathrm{Beta}(\boldsymbol{M}_+^{(n)}(a,b) + \boldsymbol{\beta}_+(a,b), \boldsymbol{M}_-^{(n)}(a,b) + \boldsymbol{\beta}_-(a,b))}{\mathrm{Beta}(\boldsymbol{\beta}_+(a,b), \boldsymbol{\beta}_-(a,b))} & \text{otherwise .}
\end{cases}
$$

where $m_a = \sum_{j \neq i} z_{j,a}$ is the size of the $a^{th}$ functional unit disregarding the assignment of the $i^{th}$ node. We note that this posterior likelihood can be efficiently calculated only considering the parts of the computation of $\boldsymbol{M}_+^{(n)}(a,b)$ and $\boldsymbol{M}_-^{(n)}(a,b)$ as well as evaluation of the Beta function that are affected by the considered assignment change.

**Scoring the functional units in terms of stability:** By sampling we obtain a large amount of potential solutions, however, for visualization and interpretation it is difficult to average across all samples as this requires that the extracted groups in different samples and runs can be related to each other. For *visualization* we instead selected the single best extracted sample $r^*$ (i.e., the MAP estimate) across 10 separate randomly initialized runs each of 500 iterations.

To facilitate interpretation we displayed the top 20 extracted functional units most reproducible across the separate runs. To identify these functional units we analyzed how often nodes co-occurred in the same cluster across the extracted samples from the other random starts $r$ according to $\boldsymbol{C} = \sum_{r \neq r^*}(\boldsymbol{Z}^{(r)}\boldsymbol{Z}^{(r)^\top} - \boldsymbol{I})$ using the following score $\eta_c$

$$\eta_c = \frac{s_c}{s_c^{tot}}, \quad s_c = \tfrac{1}{2}\boldsymbol{z}_c^{(r^*)^\top}\boldsymbol{C}\boldsymbol{z}_c^{(r^*)}, \quad s_c^{tot} = \boldsymbol{z}_c^{(r^*)^\top}\boldsymbol{C}\boldsymbol{e} - s_c.$$

$s_c$ counts the number of times the voxels in group $c$ co-occurred with other voxels in the group whereas $s_{tot}$ gives the total number of times voxels in group $c$ co-occurred with other voxels in the graph. As such $0 \leq \eta_c \leq 1$ where 1 indicates that all voxels in the $c^{th}$ group were in the same cluster across all samples whereas 0 indicates that the voxels never co-occurred in any of the other samples.

## 3 Results and Discussion

Following [11] we calculated the average shortest path length $\langle L \rangle$, average clustering coefficient $\langle C \rangle$, degree distribution $\gamma$ and largest connected component (i.e., giant component) $G$ for each subject specific graph as well as the MI threshold value $t_c$ used to define the top $100,000$ links. In table 1 it can be seen that the derived graphs are far from Erdös-Rényi random graphs. Both the clustering coefficient, degree distribution parameter $\gamma$ and giant component $G$ differ significantly from the random graphs. However, there are no significant differences between the Normal and MS group indicating that these global features do not appear to be affected by the disease.

For each run, we initialized the IRM model with $D = 50$ randomly generated functional units. We set the prior $\boldsymbol{\beta}_+(a,b) = \begin{cases} 5 & \text{a} = b \\ 1 & \text{otherwise} \end{cases}$ and $\boldsymbol{\beta}_-(a,b) = \begin{cases} 1 & \text{a} = b \\ 5 & \text{otherwise} \end{cases}$ favoring a priori higher *within functional unit link density* relative to *between link density*. We set $\alpha = \log J$ (where $J$ is the number of voxel groups). In the model estimation we treated $2.5\%$ of the links and an equivalent number of non-links as missing at random in the graphs. When treating entries as missing at random these can be ignored maintaining counts only over the observed values [16]. The estimated models are very stable as they on average extracted $D = 72.6 \pm 0.6$ functional units. In figure 2 the area under curve (AUC) scores of the receiver operator characteristic for predicting links are given for each subject where the prediction of links was based on averaging over the final 100 samples. While these AUC scores are above random for all subjects we see a high degree of variability across the subjects in terms of the model's ability to account for links and non-links in the graphs. We found no significant difference between the Normal and MS group in terms of the

Table 1: Median threshold values $t_c$, average shortest path $\langle L \rangle$, average clustering coefficient $\langle C \rangle$, degree distribution exponent $\gamma$ (i.e. $p(k) \propto k^{-\gamma}$) and giant component $G$ (i.e. largest connected component in the graphs relative to the complete graph) for the normal and multiple-sclerosis group as well as a non-parametric test of difference in median between the two groups. The random graph is an Erdös-Rényi random graph with same density as the constructed graphs.

|  | $t_c$ | $\langle L \rangle$ | $\langle C \rangle$ | $\gamma$ | $G$ |
|---|---|---|---|---|---|
| **Normal** | 0.0164 | 2.77 | 0.1116 | 1.40 | 0.8587 |
| **MS** | 0.0163 | 2.70 | 0.0898 | 1.36 | 0.8810 |
| **Random** | - | 2.73 | 0.0079 | 0.88 | 1 |
| **P-value**(Normal vs. MS) | 0.9964 | 0.4509 | 0.9954 | 0.7448 | 0.7928 |
| **P-value**(Normal and MS vs. Random) | - | 0.6764 | $p \leq 0.001$ | $p \leq 0.001$ | $p \leq 0.001$ |

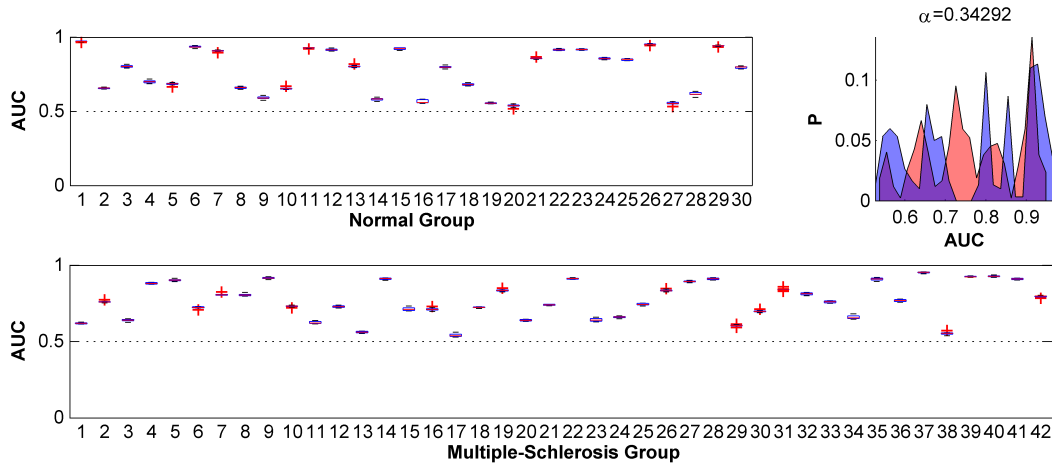

Figure 2: AUC score across the 10 different runs for each subject in the Normal group (top) and MS group (bottom). At the top right the distribution of the AUC scores is given for the two groups (Normal: blue, MS: red). No significant difference between the median value of the two distributions are found ($p \approx 0.34$).

model's ability to account for the network dynamics. Thus, there seem to be no difference in terms of how well the IRM model is able to account for structure in the networks of MS and Normal subjects. Finally, we see that the link prediction is surprisingly stable for each subject across runs as well as links and non-links treated as missing. This indicate that there is a high degree of variability in the graphs extracted from resting state fMRI between the subjects relative to the variability within each subject.

Considering the inference a stochastic optimization procedure we have visualized the sample with highest likelihood (i.e. the MAP estimate) over the runs in figure 3. We display the top 20 most reproducible extracted voxel groups (i.e., functional units) across the 10 runs. Fifteen of the 20 functional units are easily identified as functionally relevant networks. These selected functional units are similar to the networks previously identified on resting-state fMRI data using ICA [9]. The sensori-motor network is represented by the functional units 2, 3, 13 and 20; the posterior part of the default-mode network [19] by functional units 6, 14, 16, 19; a fronto-parietal network by the functional units 7,10 and 12; the visual system represented by the functional units 5, 11, 15, 18. Note the striking similarity to the sensori-motor ICA1, posterior part of the default network ICA2 and fronto-parietal network ICA3 and visual component ICA4. Contrary to ICA the current approach is able to also model interactions between components and a consistent pattern is revealed where the functional units with the highest within connectivity also show the strongest between connectivity. Furthermore the functional units appear to have symmetric connectivity profiles e.g. functional unit 2 is strongly connected to functional unit 3 (sensori-motor system), and these both strongly connect to the same other functional units, in this case 6 and 16 (default-mode network). Functional units 1, 4, 8, 9, 17 we attribute to vascular noise and these units appear to be less connected with the remaining functional units.

In panel C of figure 3 we tested the difference between medians in the connectivity of the extracted functional units. Given are connections that are significant at $p \leq 0.05$. Healthy individuals show stronger connectivity among selected functional units relative to patients. The functional units involved are distributed throughout the brain and comprise the visual system (functional unit 5 and 11), the sensori-motor network (functional unit 2), and the fronto-parietal network (functional unit 10). This is expected since MS affects the brain globally by white-matter changes disseminated throughout the brain [12]. Patients with MS show stronger connectivity relative to healthy individuals between selected parts of the sensori-motor (functional unit 13) and fronto-parietal network (functional units 7 and 12). An interpretation of this finding could be that the communication increases between the fronto-parietal and the sensori-motor network either as a maladaptive consequence of the disease or as part of a beneficial compensatory mechanism to maintain motor function.

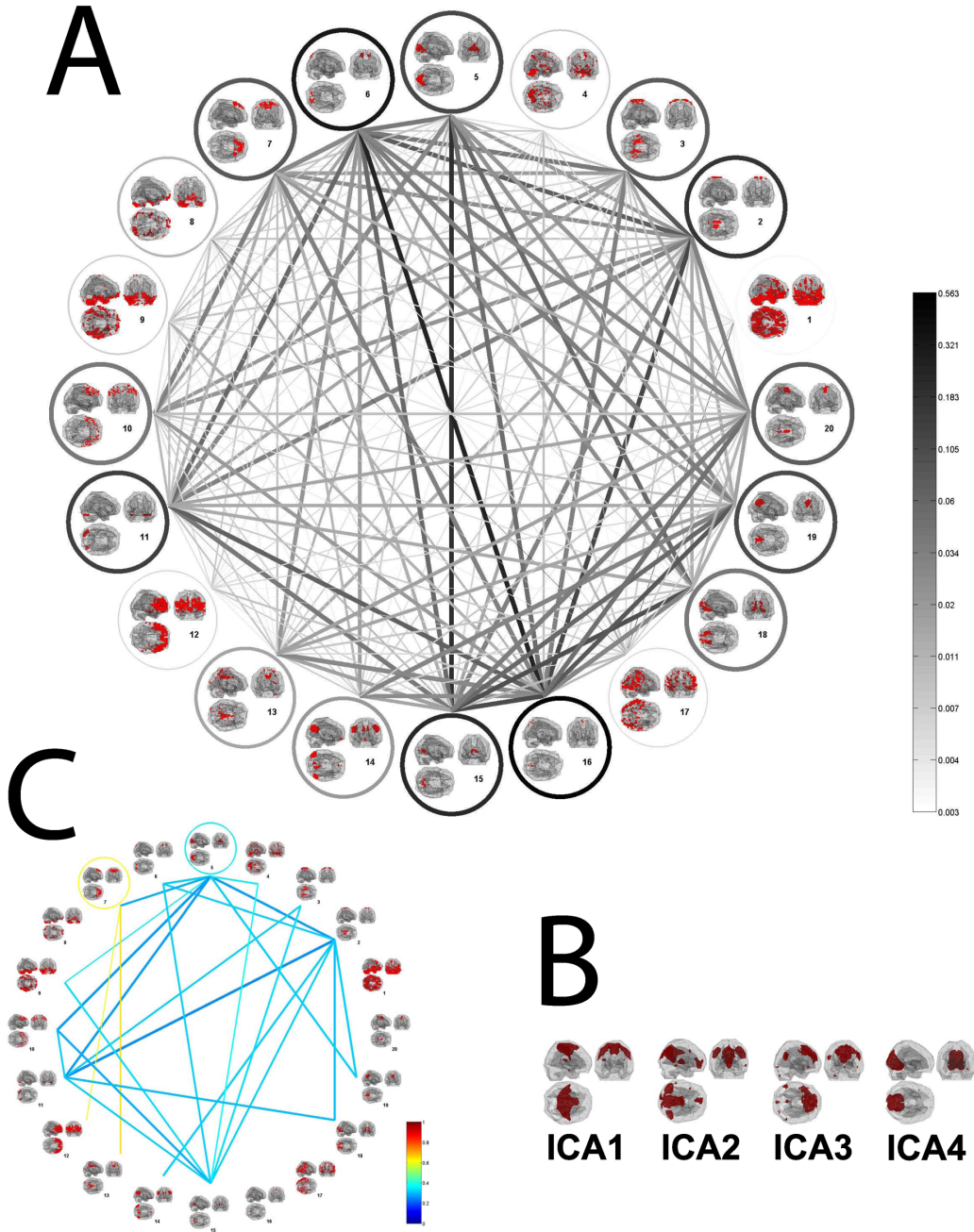

Figure 3: **Panel A:** Visualization of the MAP model over the 10 restarts. Given are the functional units indicated in red while circles indicate median within unit link density and lines median between functional unit link density. Gray scale and line width code the link density between and within the functional units using a logarithmic scale. **Panel B:** Selected resting state components extracted from a group independent component analysis (ICA) are given. After temporal concatenation over subjects the Infomax ICA algorithm [2] was used to identify 20 spatially independent components. Subsequently the individual component time series was used in a regression model to obtain subject specific component maps [5]. The displayed ICA maps are based on one sample t-tests corrected for multiple comparisons $p \leq 0.05$ using Gaussian random fields theory. **Panel C:** AUC score for relations between the extracted groups thresholded at a significance level of $\alpha = 5\%$ based on a two sided rank-sum test. Blue indicates that the link density is larger for Normal than MS, yellow that MS is larger than Normal. (A high resolution version of the figure can be found in the supplementary material).

Table 2: Leave one out classification performance based on support vector machine (SVM) with a linear kernel, linear discriminant analysis (LDA) and K-nearest neighbor (KNN). Significance level estimated by comparing to classification performance for the corresponding classifiers with randomly permuted class labels, bold indicates significant classification at a $p \leq 0.05$.

|  | Raw data | PCA | ICA | Degree | IRM |
|---|---|---|---|---|---|
| **SVM** | 51.39 | 55.56 | **63.89** ($p \leq 0.04$) | 59.72 | **72.22**($p \leq 0.002$) |
| **LDA** | 59.72 | 51.39 | **63.89** ($p \leq 0.05$) | 51.39 | **75.00**($p \leq 0.001$) |
| **KNN** | 38.89 | 58.33 | 56.94 | 51.39 | **66.67**($p \leq 0.01$) |

**Discriminating Normal subjects from MS:** We evaluated the classification performance of the subject specific group link densities $\rho^{(n)}$ based on leave one out cross-validation. We considered three standard classifiers, soft margin support vector machine (SVM) with linear kernel ($C = 1$), linear discriminant analysis (LDA) based on the pooled variance estimate (features projected by principal component analysis to a 20 dimensional feature space prior to analysis), as well as K-nearest neighbor (KNN), $K = 3$. We compared the classifier performances to classifying the normalized raw subject specific $voxel \times time$ series, i.e. the matrix given by $subject \times voxel - time$ as well as the data projected to the most dominant 20 dimensional subspace denoted (PCA). For comparison we also included a group ICA [5] analysis as well as the performance using node degree (Degree) as features which has previously been very successful for classification of schizophrenia [6]. For the IRM model we used the Bayesian average over predictions which was dominated by the MAP estimate given in figure 3. For all the classification analyses we normalized each feature. In table 2 is given the classification results. Group ICA as well as the proposed IRM model significantly classify above random. The IRM model has a higher classification rate and is significant across all the classifiers.

Finally, we note that contrary to analysis based on temporal correlation such as the ICA and PCA approaches used for the classification the benefit of mutual information is that it can take higher order dependencies into account that are not necessarily reflected by correlation. As such, a brain region driven by the variance of another brain region can be captured by mutual information whereas this is not necessarily captured by correlation.

## 4   Conclusion

The functional units extracted using the IRM model correspond well to previously described RSNs [19, 9]. Whereas conventional models for assessing functional connectivity in rs-fMRI data often aim to divide the brain into segregated networks the IRM explicitly models relations between functional units enabling visualization and analysis of interactions. Using classification models to predict the subject disease state revealed that the IRM model had a higher prediction rate than discrimination based on the components extracted from a conventional group ICA approach [5]. IRM readily extends to directed graphs and networks derived from task related functional activation. As such we believe the proposed method constitutes a promising framework for the analysis of functionally derived brain networks in general.

## References

[1] S. Achard, R. Salvador, B. Whitcher, J. Suckling, and E. Bullmore. A resilient, low-frequency, small-world human brain functional network with highly connected association cortical hubs. *The Journal of Neuroscience*, 26(1):63–72, 2006.

[2] A. J. Bell and T. J. Sejnowski. An information maximization approach to blind source separation and blind deconvolution. *Neural Computation*, 7:1129–1159, 1995.

[3] B. Biswal, F. Z. Yetkin, V. M. Haughton, and J. S. Hyde. Functional connectivity in the motor cortex of resting human brain using echo-planar MRI. *Magnetic Resonance in Medicine*, 34(4):537–541, 1995.

[4] E. Bullmore and O. Sporns. Complex brain networks: graph theoretical analysis of structural and functional systems. *Nature Reviews. Neuroscience*, 10(3):186-98, 2009.

[5] V. D. Calhoun, T. Adali, G. D. Pearlson, and J. J. Pekar. A method for making group inferences from functional MRI data using independent component analysis. *Human Brain Mapping*, 14:140–151, 2001.

[6] G. Cecchi, I. Rish, B. Thyreau, B. Thirion, M. Plaze, M.-L. Paillere-Martinot, C. Martelli, J.-L. Martinot, and J.-B. Poline. Discriminative network models of schizophrenia. *Advances in Neural Information Processing Systems*, 22:252–260, 2009.

[7] B. Chai, D. Walther, D. Beck, and L. Fei-Fei. Exploring functional connectivities of the human brain using multivariate information analysis. *Advances in Neural Information Processing Systems*, 22:270–278, 2009.

[8] D. B. Dahl. Sequentially-allocated merge-split sampler for conjugate and nonconjugate Dirichlet process mixture models. Technical report, Texas A&M University, 2005.

[9] J. S. Damoiseaux, S.A.R.B. Rombouts, F. Barkhof, P. Scheltens, C. J. Stam, S. M. Smith, and C. F. Beckmann. Consistent resting-state networks across healthy subjects. *Proceedings of the National Academy of Sciences of the United States of America*, 103(37):13848–13853, 2006.

[10] M. De Luca, S. Smith, N. De Stefano, A. Federico, and P. M. Matthews. Blood oxygenation level dependent contrast resting state networks are relevant to functional activity in the neocortical sensorimotor system. *Experimental Brain Research*, 167(4):587–594, 2005.

[11] V. M. Eguiluz, D. R. Chialvo, G. A. Cecchi, M. Baliki, and A. V. Apkarian. Scale-free brain functional networks. *Physical Review Letters*, 94(1):018102, 2005.

[12] M. Filippi and M. A. Rocca. MRI evidence for multiple sclerosis as a diffuse disease of the central nervous system. *Journal of Neurology*, 252 Suppl 5:v16–v24, 2005.

[13] K.J. Friston, L. Harrison, and W.D. Penny. Dynamic Causal Modelling. *NeuroImage*, 19(4):1273–1302, 2003.

[14] G. H. Glover, T. Q. Li, and D. Ress. Image-based method for retrospective correction of physiological motion effects in fMRI: RETROICOR. *Magnetic Resonance in Medicine*, 44:162–167, 2000.

[15] S. Jain and R. M. Neal. A split-merge markov chain monte carlo procedure for the dirichlet process mixture model. *Journal of Computational and Graphical Statistics*, 13(1):158–182, 2004.

[16] C. Kemp, J. B. Tenenbaum, T. L. Griffiths, T. Yamada, and N. Ueda. Learning systems of concepts with an infinite relational model. In *Artificial Intelligence, Proceedings of the 21st National AAAI Conference on*, 1:381–388, 2006.

[17] K. Nowicki and T. A. B. Snijders. Estimation and prediction for stochastic blockstructures. *Journal of the American Statistical Association*, 96(455):1077–1087, 2001.

[18] J. W. Peterson, L. Bö, S. Mörk, A. Chang, and B. D. Trapp. Transected neurites, apoptotic neurons, and reduced inflammation in cortical multiple sclerosis lesions. *Annals of Neurology*, 50:389–400, 2001.

[19] M. E. Raichle, A. M. MacLeod, A. Z. Snyder, W. J. Powers, D. A. Gusnard, and G. L. Shulman. A default mode of brain function. *Proceedings of the National Academy of Sciences of the United States of America*, 98(2):676–682, 2001.

[20] A. J. Storkey, E. Simonotto, H. Whalley, S. Lawrie, L. Murray, and D. McGonigle. Learning structural equation models for fMRI. *Advances in Neural Information Processing Systems*, 19:1329–1336, 2007.

[21] B. D. Trapp, J. Peterson, R. M. Ransohoff, R. Rudick, S. Mörk, and L Bö. Axonal transection in the lesions of multiple sclerosis. *The New England journal of medicine*, 338(5):278–85, 1998.

[22] A. Tsai, J. W. Fisher, III, C. Wible, W. M. Wells, III, J. Kim, and A. S. Willsky. Analysis of functional MRI data using mutual information. In *MICCAI '99: Proc. of the Sec. Intern. Conf. on Medical Image Computing and Computer-Assisted Intervention, Lecture Notes in Computer Science*, 1679:473–480, 1999.

[23] N. Tzourio-Mazoyer, B. Landeau, D. Papathanassiou, F. Crivello, O. Etard, N. Delcroix, B. Mazoyer, and M. Joliot. Automated anatomical labeling of activations in SPM using a macroscopic anatomical parcellation of the MNI MRI single-subject brain. *NeuroImage*, 15(1):273–89, 2002.

[24] M. van den Heuvel, R. Mandl, and H. Hulshoff Pol. Normalized cut group clustering of resting-state fMRI data. *PLoS ONE*, 3(4), 2008.

[25] J. Walters-williams, Y. Li. Estimation of Mutual Information: A Survey. *Lecture Notes in Computer Science*, 5589:389–396, 2009.

[26] S. G Waxman. Axonal conduction and injury in multiple sclerosis: the role of sodium channels. *Nature reviews. Neuroscience*, 7(12):932–41, 2006.

[27] Z. Xu, V. Tresp, K. Yu, and H. P. Kriegel. Infinite hidden relational models. In *In Proceedings of the 22nd International Conference on Uncertainty in Artificial Intelligence*, 2006.

[28] L. Zhang, D. Samaras, N. Alia-klein, N. Volkow, and R. Goldstein. Modeling neuronal interactivity using dynamic bayesian networks. *Advances in Neural Information Processing Systems*, 18:1593–1600, 2006.

